# Phase Synchrony Rate for the Recognition of Motor Imagery in Brain-Computer Interface

**Le Song**
Nation ICT Australia
School of Information Technologies
The University of Sydney
NSW 2006, Australia
lesong@it.usyd.edu.au

**Evian Gordon**
Brain Resource Company
Scientific Chair, Brain Dynamics Center
Westmead Hospital
NSW 2006, Australia
eviang@brainresource.com

**Elly Gysels**
Swiss Center for Electronics and Microtechnology
Neuchâtel, CH-2007 Switzerland
elly.gysels@csem.ch

## Abstract

Motor imagery attenuates EEG $\mu$ and $\beta$ rhythms over sensorimotor cortices. These amplitude changes are most successfully captured by the method of Common Spatial Patterns (CSP) and widely used in brain-computer interfaces (BCI). BCI methods based on amplitude information, however, have not incorporated the rich phase dynamics in the EEG rhythm. This study reports on a BCI method based on phase synchrony rate (SR). SR, computed from binarized phase locking value, describes the number of discrete synchronization events within a window. Statistical nonparametric tests show that SRs contain significant differences between 2 types of motor imageries. Classifiers trained on SRs consistently demonstrate satisfactory results for all 5 subjects. It is further observed that, for 3 subjects, phase is more discriminative than amplitude in the first 1.5-2.0 s, which suggests that phase has the potential to boost the information transfer rate in BCIs.

## 1 Introduction

A brain-computer interface (BCI) is a communication system that relies on the brain rather than the body for control and feedback. Such an interface offers hope not only for those severely paralyzed to control wheelchairs but also to enhance normal performance. Current BCI research is still in its infancy. Most studies focus on finding useful brain signals and designing algorithms to interpret them [1, 2].

The most exploited signal in BCI is the scalp-recorded electroencephalogram (EEG). EEG is a noninvasive measurement of the brain's electrical activities and has a temporal resolution of milliseconds. It is well known that motor imagery attenuates EEG $\mu$ and $\beta$ rhythm over sensorimotor cortices. Depending on the part of the body imagined moving, the am-

plitude of multichannel EEG recordings exhibits distinctive spatial patterns. Classification of these patterns is used to control computer applications. Currently, the most successful method for BCI is called Common Spatial Patterns (CSP). The CSP method constructs a few new time series whose variances contain the most discriminative information. For the problem of classifying 2 types of motor imageries, the CSP method is able to correctly recognize 90% of the single trials in many studies [3, 4]. Ongoing research on the CSP method mainly focuses on its extension to the multi-class problem [5] and its integration with other forms of EEG amplitude information [4].

EEG signals contain both amplitude and phase information. Phase, however, has been largely ignored in BCI studies. Literature from neuroscience suggests, instead, that phase can be more discriminative than amplitude [6, 7]. For example, compared to a stimuli in which no face is present, face perception induces significant changes in $\gamma$ synchrony, but not in amplitude [6]. Phase synchrony has been proposed as a mechanism for dynamic integration of distributed neural networks in the brain. Decreased synchrony, on the other hand, is associated with active unbinding of the neural assemblies and preparation of the brain for the next mental state (see [7] for a review). Accumulating evidence from both micro-electrode recordings [8,9] and EEG measurements [6] provides support to the notion that phase dynamics subserve all mental processes, including motor planning and imagery.

In the BCI community, only a paucity of results has demonstrated the relevance of phase information [10–12]. Fewer studies still have ever compared the difference between amplitude and phase information for BCI. To address these deficits, this paper focuses on three issues:

- Does binarized phase locking value (PLV) contain relevant information for the classification of motor imageries?
- How does the performance of binarized PLV compare to that of non-binarized PLV?
- How does the performance of methods based on phase information compare to that of the CSP method?

In the remainder of the paper, the experimental paradigm will be described first. The details of the method based on binarized PLV are presented in Section 3. Comparison between PLV, binarized PLV and CSP are then made in Section 4. Finally, conclusions are provided in Section 5.

## 2  Recording paradigm

Data set IVa provided by the Berlin BCI group [5] is investigated in this paper (available from the BCI competition III web site). Five healthy subjects (labeled 'aa', 'al', 'av', 'aw' and 'ay' respectively) participated in the EEG recordings. Based on the visual cues, they were required to imagine for 3.5 s either right hand (type 1) or right foot movements (type 2). Each type of motor imagery was carried out 140 times, which results in 280 labeled trials for each subject. Furthermore, the down-sampled data (at 100 Hz) is used. For the convenience of explanation, the length of the data is also referred to as time points. Therefore, the window for the full length of a trial is [1, 350].

## 3  Feature from phase

### 3.1  Phase locking value

Two EEG signals $x_i(t)$ and $x_j(t)$ are said to be synchronized, if their instantaneous phase difference $\psi_{ij}(t)$ (complex-valued with unit modulus) stays constant for a period of time

$\Delta_\psi$. Phase locking value (PLV) is commonly used to quantify the degree of synchrony, i.e.

$$PLV_{ij}(t) = \frac{1}{\Delta_\psi} \left| \sum_{t-\Delta_\psi}^{t} \psi_{ij}(t) \right| \in [0, 1], \qquad (1)$$

where 1 represents perfect synchrony. The instantaneous phase difference $\psi_{ij}(t)$ can be computed using either wavelet analysis or Hilbert transformation. Studies show that these two approaches are equivalent for the analysis of EEG signals [13]. In this study, Hilbert transformation is employed in a similar manner to [10].

### 3.2 Synchrony rate

Neuroscientists usually threshold the phase locking value and focus on statistically significant periods of strong synchrony. Only recently, researchers begin to study the transition between high and low levels of synchrony [6, 14, 15]. Most notably, the researcher in [15] transformed PLV into discrete values called link rates and showed that link rates could be a sensitive measure to relevant changes in synchrony. To investigate the usefulness of discretization for BCIs, we binarize the time series of PLV and define synchrony rate based on them.

The threshold chosen to binarize PLV minimizes the quantization error. Suppose that the distribution of PLV is $p(x)$, then the threshold $th_0$ is determined by

$$th_0 = \arg\min_{th} \int_0^1 (x - g(x - th))^2 \, p(x) \mathrm{d}x, \qquad (2)$$

where $g(\cdot)$ is the hard-limit transfer function which assumes 1 for non-negative numbers and 0 otherwise. In practice, $p(x)$ is computed at discrete locations and the integration is replaced by summation. For the data set investigated, $th_0$s are similar across 5 subjects ($\simeq$ 0.5) when EEG signals are filtered between 4 and 40Hz and $\Delta_\psi$ is 0.25 s (These parameters are used in the Result section for all 5 subjects). The thresholded sequences are binary and denoted by $b_{ij}(t)$.

The ones in $b_{ij}(t)$ can be viewed as discrete events of strong synchrony, while zeros are those of weak synchrony. The resemblance of $b_{ij}(t)$ to the spike trains of neurons prompts us to define synchrony rate (SR)—the number of discrete events of strong synchrony per second. Formally, given a window $\Delta_b$, the synchrony rate $r_{ij}(t)$ at time $t$ is:

$$r_{ij}(t) = \frac{1}{\Delta_b} \sum_{t-\Delta_b}^{t} b_{ij}(t). \qquad (3)$$

SR describes the average level of synchrony between a pair of electrodes in a given window. The size of the window will affect the value of the SR. In the next section, we will detail the choice of the windows and the selection of features from SRs.

### 3.3 Feature extraction

Before computing synchrony rates, a circular Laplacian [16] is applied to boost the spatial resolution of the raw EEG. This method first interpolates the scalp EEG, and then re-references EEG using interpolated values on a circle around an electrode. Varying the radius of the circles achieves different spatial filtering effects, and the best radius is tuned for individual subject.

Spatially filtered EEG is split into 6 sliding windows of length 100, namely [1, 100], [51, 150], [101, 200], [151, 250], [201, 300] and [251, 350]. Each window is further divivded

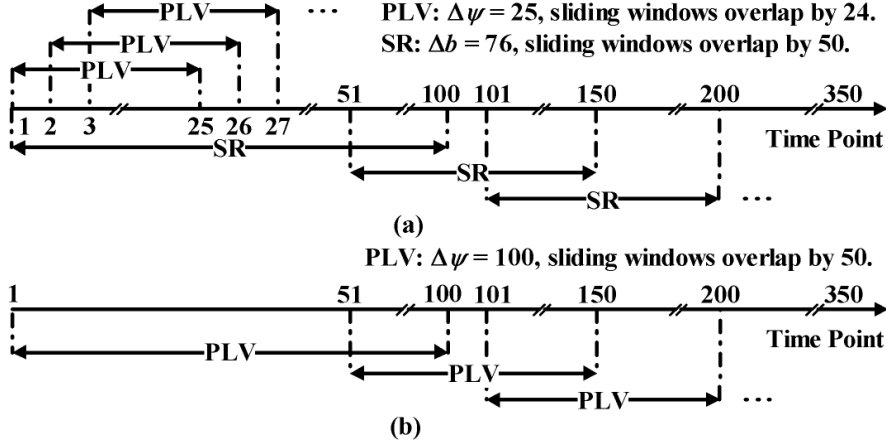

Figure 1: Overall scheme of window division for (a) the synchrony rate (SR) method and (b) the phase locking value (PLV) method. $\Delta_\psi$ for the SR method covers the length of a micro-window, while that for the PLV method corresponds to the length of a sliding window. $\Delta_b$ is equal to $100 - \Delta_\psi + 1$. (Note: time axis is NOT uniformly scaled.)

into 76 micro-windows (with size 25 and overlap by 24). PLVs are then computed and binarized for each micro-window (according to (1)). Averaging the 76 binarized PLVs results in the SR (according to (3)). As a whole, 6 SRs will be computed for each electrode pair in a trial. SRs from all electrode pairs will be passed to statistical tests and further used as features for classification. The overall scheme of this window division is illustrated in Fig 1(a). In order to compare PLV and SR, PLVs are also computed for the full length of each sliding window (Fig. 1(b)), which results in 6 PLVs for each electrode pair. These PLVs will go through the same statistical tests and classification stage.

### 3.4 Statistical test

A key observation is that both PLVs and SRs contain many statistically significant differences between the 2 types of motor imagery in almost every sliding window. Statistical nonparametric tests [17] are employed to locate these differences. For each electrode pair, a null hypothesis—$H_0$: The difference of the mean SR/PLV for the 2 types of motor imagery is zero—is formulated for each sliding window. Then the distribution of the difference is obtained by 1000 randomization. The hypothesis is rejected if the difference of the original data is larger than 99.5% or smaller than 0.5% of those from randomized data (equivalent to $p < 0.01$).

Fig. 2 illustrates the test results with data from subject 'av'. For simplicity, only those SRs with significant increase are displayed. Although the exact locations of these increases differ from window to window, some general patterns can be observed. Roughly speaking, window 2, 3 and 4 can be grouped as similar, while window 1, 5 and 6 are different from each other. Window 1 reflects changes in the early stage of a motor imagery, consisting increased couplings mainly within visual cortices and between visual and motor cortices. Then (window 2, 3 and 4) increased couplings occur between motor cortices of both hemispheres and between lateral and mesial areas of the motor cortices. During the last stage, these couplings first (window 5) shift to the left hemisphere and then (window 6) reduce to some sparse distant interactions. Similar patterns can also be discovered from the PLVs (not illustrated). Although the exact functional interpretation of these patterns awaits further investigation, they can be treated as potential features for classification.

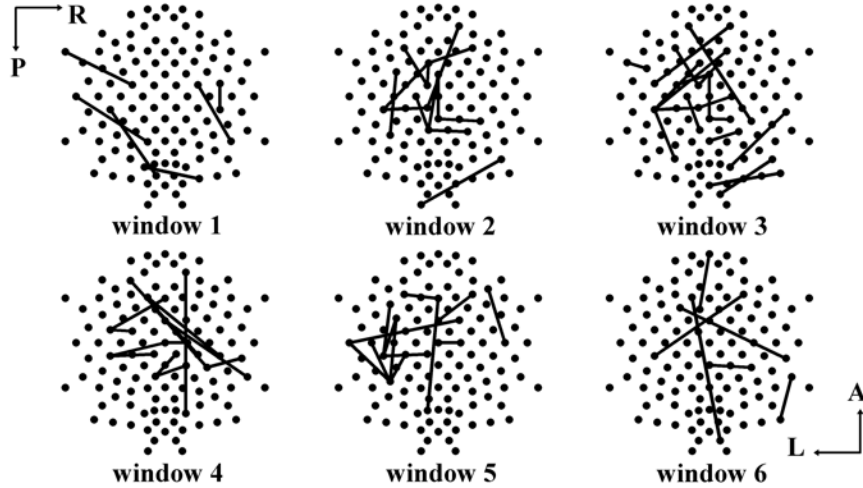

Figure 2: Significantly increased synchrony rates in right hand motor imagery. Data are from subject 'av'. (A: anterior; L: left; P: posterior; R: right.)

## 4 Classification strategy

To evaluate the usefulness of synchrony rate for the classification of motor imagery, $50 \times 2$-fold cross validation is employed to compute the generalization error. This scheme randomizes the order of the trials for 50 times. Each randomization further splits the trials into two equal halves (of 70 trials), each serving as training data once. There are four steps in each fold. Averaging the prediction errors from each fold results in the generalization error.

• Compute SRs for each trial (including both training and test data). As illustrated in Fig. 1(a), this results in a 6 dimensional (one for each window) feature vector for each electrode pair ($6903 = \frac{118 \times (118-1)}{2}$ pairs in total). Alternatively, it can be viewed as a 6903 dimensional feature vector for each window.

• Filter features using the Fisher ratio. The Fisher ratio (a variant, $\frac{|\mu_+ - \mu_-|}{\sigma_+ + \sigma_-}$, is used in the actual computation) measures the discriminability of individual feature for classification task. It is computed using training data only, and then compared to a threshold (0.3), below which a feature is discarded. The indices of the selected features are further used to filter the test data. The selected features are not necessarily all those located by the statistical tests. Generally, they are only a subset of the most significant SRs.

• Train a linear SVM for each window and use meta-training scheme to combine them. The evolving nature of the SRs (illustrated in Fig. 2) suggests that information in the 6 windows may be complementary to each other. Similar to [4], a second level of linear SVM is trained on the output of the SVMs for individual windows. This meta-training scheme allows us to exploit the inter-window relations. (Note that this step is carried out strictly on the training data.)

• Predict the label of the test data. Test data are fed into the two-level SVM, and the prediction error is measured as the proportion of the misclassified trials.

The above four steps are also used to compute the generalization errors for the PLV method. The only modification is in step one, where PLVs are computed instead of SRs (Fig. 1(b)). In the next section, we will present the generalization errors for both SR and PLV method, and compare them to those of the CSP method.

# 5 Result and comparison

## 5.1 Generalization error

Table 1 shows the generalization errors in percentage (with standard deviations) for both synchrony rate and PLV method. For comparison, we also computed the generalization errors of the CSP method [3] using linear SVM and $50 \times 2$-fold cross validation. The parameters of the CSP method (including filtering frequency, the number of channels used and the number of spatial patterns selected) are individually tuned for each subject according the competition winning entry of data set IVa [18]. Note that all errors in Table 1 are computed using the full length (3.5 s) of a trial.

Generally, the errors of the SR method is higher than those of the PLV method. This is because SR is an approximation of PLV by definition. Remember that during the computation of SRs, the PLVs in the micro-windows are first binarized with a threshold $th_0$. This threshold is so chosen that the approximation is as close to its original as possible. It works especially well for two of the subjects ('al' and 'ay'), with the difference between the two methods less than 1%. Although SR method produces higher errors, it may have some advantages in practice. Especially for hardware implemented BCI systems, smaller window for PLV computation means smaller buffer and binarized PLV makes further processing easier and faster.

The errors of the CSP method is lowest for most of the subjects. For subject 'aa' and 'aw', it is better than the other two methods by 10-20%, but the gaps are narrowed for subject 'al' and 'av' (less than 2.5%). Most notably, for subject 'ay', the SR method even outperforms the CSP method by about 5%. Remember that the CSP method is implemented using individually optimized parameters, while those for the SR and PLV method are the same across the 5 subjects. Fine tuning the parameters has the potential to further improve the performance of the latter two methods. The errors computed above, however, reveals only partial difference between the three methods. In the next subsection, a more thorough investigation will be carried out using information transfer rates.

## 5.2 Information transfer rate

Information transfer rate (ITR) [1] is the amount of information (measured in bits) generated by a BCI system within a second. It takes both the error and the length of a trial into account. If two BCI systems produce the same error, the one with a short trial will have higher information transfer rate. To investigate the performance of the three methods in this context, we shortened the trials into 5 different lengths, namely 1.0 s, 1.5 s, 2.0 s, 2.5 s and 3.0 s. The generalization errors are computed for these shortened trials and then converted into information transfer rates, as showed in Fig. 3.

Interesting results emerge from the curves in Fig. 3. Most subjects (except subject 'aw') achieve the highest information transfer rates within the first 1.5-2.0 s. Although longer trials usually decrease the errors, they do not necessarily result in increased information transfer rates. Furthermore, for subject 'al', 'av' and 'ay', the highest information transfer

Table 1: Generalization errors (%) of the synchrony rate (SR), PLV and CSP methods

| Subject | aa | al | av | aw | ay |
|---------|-----|-----|-----|-----|-----|
| **SR** | 29.34±3.97 | 4.05±1.28 | 32.67±3.41 | 22.96±4.39 | 5.93±1.75 |
| **PLV** | 23.05±3.39 | 3.59±1.28 | 29.91±3.23 | 18.65±3.48 | 5.41±1.53 |
| **CSP** | 12.58±2.56 | 2.65±1.35 | 30.30±3.02 | 3.16±1.32 | 11.43±2.34 |

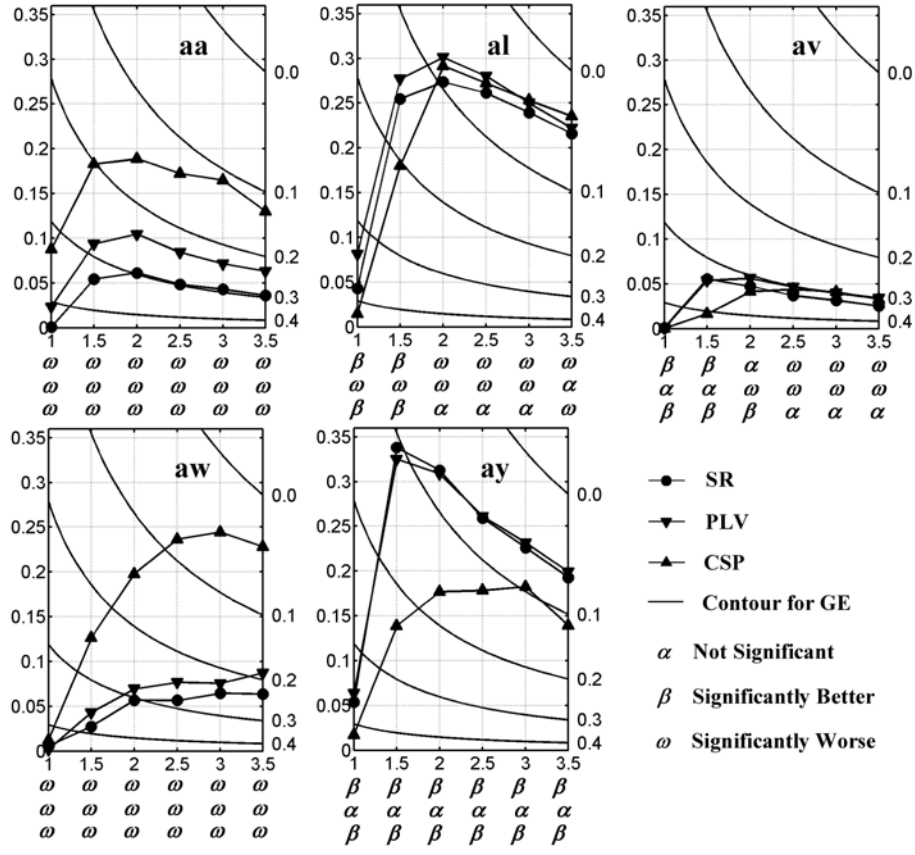

Figure 3: Information transfer rates (ITR) for synchrony rate (SR), PLV and CSP method. Horizontal axis is time T (in seconds). Vertical axis on the left measures information transfer rate (in bit/second) and that on the right shows the generalization error (GE) in decimals. The three lines of Greek characters under each subplot code the results of statistical comparisons (Student's t-test, significance level 0.01) of different methods. Line 1 is the comparison between SR and CSP methods; Line 2 is between SR and PLV method; and Line 3 between PLV and CSP method.

rates are achieved by methods based on phase. Especially for subject 'ay', phase generates about 0.2 bits more information per second. The qualitative similarity between SR and PLV method suggests that phase can be more discriminative than amplitude within the first 1.5-2.0 s. Common to the three methods, however, the near zero information transfer rates within the first second virtually pose a limit for BCIs. In the case where real-time application is of high priority, such as navigating wheelchairs, this problem is even more pronounced. Incorporating phase information and continuing the search for new features have the potential to overcome this limit.

## 6 Conclusion

EEG phase contains complex dynamics. Changes of phase synchrony provide complementary information to EEG amplitude. Our results show that within the first 1.5-2.0 s of a motor imagery, phase can be more useful for classification and can be exploited by our synchrony rate method. Although methods based on phase have achieved good results in

some subjects, the subject-wise difference and the exact functional interpretation of the selected features need further investigation. Solving these problems have the potential to boost information transfer rates in BCIs.

## Acknowledgments

The author would like to thank Ms. Yingxin Wu and Dr. Julien Epps from NICTA, and Dr. Michael Breakspear from Brain Dynamics Center for discussion.

## References

[1] J.R. Wolpaw et al., "Brain-computer interface technology: a review of the first international meeting," *IEEE Trans. Rehab. Eng.*, vol. 8, pp. 164-173, 2000.

[2] T.M. Vaughan et al., "Brain-computer interface technonolgy: a review of the second international meeting," *IEEE Trans. Rehab. Eng.*, vol. 11, pp. 94-109, 2003.

[3] H. Ramoser, J. Müller-Gerking, and G. Pfurtscheller, "Optimal spatial filtering of single trial EEG during imagined hand movement," *IEEE Trans. Rehab. Eng.*, vol. 8, pp. 441-446, 2000.

[4] G. Dornhege, B. Blankertz, G. Curio, and K.R. Müller, "Combining features for BCI," *Advances in Neural Inf. Proc. Systems (NIPS 02)*, vol. 15, pp. 1115-1122, 2003.

[5] G. Dornhege, B. Blankertz, G. Curio, and K.R. Müller, "Boosting bit rates in non-invasive EEG single-trial classifications by feature combination and multi-class paradigms," *IEEE Trans. Biomed. Eng.*, vol. 51, pp. 993-1002, 2004.

[6] E. Rodriguez et al., "Perception's shadow: long distance synchronization of human brain activity," *Nature*, vol. 397, pp. 430-433, 1999.

[7] F. Varela, J.P. Lachaux, E. Rodriguez, and J. Martinerie, "The brainweb: phase synchronization and large-scale integration," *Nature Reviews Neuroscience*, vol. 2, pp. 229-239, 2001.

[8] W. Singer, and C.M. Gray, "Visual feature integration and the temporal correlation hypothesis," *Annu. Rev. Neurosci*, vol. 18, pp. 555-586, 1995.

[9] P.R. Roelfsema, A.K. Engel, P. König, and W. Singer, "Visuomotor integration is associated with zero time-lag synchronization among cortical areas," *Nature*, vol. 385, pp. 157-161, 1997.

[10] E. Gysels, and P. Celka, "Phase synchronization for the recognition of mental tasks in brain-computer interface," *IEEE Trans. Neural Syst. Rehab. Eng.*, vol. 12, pp. 406-415, 2004.

[11] C. Brunner, B. Graimann, J.E. Huggins, S.P Levine and G. Pfurtscheller, "Phase relationships between different subdural electrode recordings in man," *Neurosci. Lett.*, vol. 275, pp.69-74, 2005.

[12] L. Song, "Desynchronization network analysis for the recognition of imagined movement in BCIs,", *Proc. of 27th IEEE EMBS conference*, Shanghai, China, September 2005.

[13] M. Le Van Quyen et al., "Comparison of Hilbert transform and wavelet methods for the analysis of neuronal synchrony," *J. Neurosci. Methods*, vol. 111, pp. 83-98, 2001.

[14] M. Breakspear, L. Williams, and C.J. Stam, "A novel method for the topographic analysis of neural activity reveals formation and dissolution of 'dynmaic cell assemblies'," *J. Comput. Neurosci.*, vol. 16, pp. 49-68, 2004.

[15] M.J.A.M van Putten, "Proposed link rates in the human brain," *J. of Neurosci. Methods*, vol. 127, pp. 1-10, 2003.

[16] L. Song, and J. Epps, "Improving separability of EEG signals during motor imagery with an efficient circular Laplacian," in preparation.

[17] T.E. Nichols, and A.P. Holmes, "Nonparametric permutation tests for functional neuroimaging: a primer with examples," *Human Brain Mapping*, vol. 15, pp. 1-25, 2001.

[18] Y.J. Wang, X.R. Gao, Z.G. Zhang, B. Hong, and S.K. Gao, "BCI competition III—data set IVa: classifying single-trial EEG during motor imagery with a small training set," *IEEE Trans. Neural Syst. Rehab. Eng.*, submitted.